# Distributed Inference for Latent Dirichlet Allocation

**David Newman, Arthur Asuncion, Padhraic Smyth, Max Welling**
Department of Computer Science
University of California, Irvine
{newman,asuncion,smyth,welling}@ics.uci.edu

## Abstract

We investigate the problem of learning a widely-used latent-variable model – the Latent Dirichlet Allocation (LDA) or "topic" model – using distributed computation, where each of $P$ processors only sees $1/P$ of the total data set. We propose two distributed inference schemes that are motivated from different perspectives. The first scheme uses local Gibbs sampling on each processor with periodic updates—it is simple to implement and can be viewed as an approximation to a single processor implementation of Gibbs sampling. The second scheme relies on a hierarchical Bayesian extension of the standard LDA model to directly account for the fact that data are distributed across $P$ processors—it has a theoretical guarantee of convergence but is more complex to implement than the approximate method. Using five real-world text corpora we show that distributed learning works very well for LDA models, i.e., perplexity and precision-recall scores for distributed learning are indistinguishable from those obtained with single-processor learning. Our extensive experimental results include large-scale distributed computation on 1000 virtual processors; and speedup experiments of learning topics in a 100-million word corpus using 16 processors.

## 1   Introduction

Very large data sets, such as collections of images, text, and related data, are becoming increasingly common, with examples ranging from digitized collections of books by companies such as Google and Amazon, to large collections of images at Web sites such as Flickr, to the recent Netflix customer recommendation data set. These data sets present major opportunities for machine learning, such as the ability to explore much richer and more expressive models, as well as providing new and interesting domains for the application of learning algorithms.

However, the scale of these data sets also brings significant challenges for machine learning, particularly in terms of computation time and memory requirements. For example, a text corpus with 1 million documents, each containing 1000 words on average, will require approximately 12 Gbytes of memory to store the $10^9$ words, which is beyond the main memory capacity for most single processor machines. Similarly, if one were to assume that a simple operation (such as computing a probability vector over categories using Bayes rule) would take on the order of $10^{-6}$ sec per word, then a full pass through $10^9$ words will take 1000 seconds. Thus, algorithms that make multiple passes over this sized corpus (such as occurs in many clustering and classification algorithms) will have run times in days.

An obvious approach for addressing these time and memory issues is to distribute the learning algorithm over multiple processors [1, 2, 3]. In particular, with $P$ processors, it is somewhat trivial to get around the memory problem by distributing $\frac{1}{P}$ of the total data to each processor. However, the computation problem remains non-trivial for a fairly large class of learning algorithms, namely how to combine local processing on each of the $P$ processors to arrive at a useful global solution.

In this general context we investigate distributed learning algorithms for the LDA model [4]. LDA models are arguably among the most successful recent learning algorithms for analyzing count data such as text. However, they can take days to learn for large corpora, and thus, distributed learning would be particularly useful for this type of model.

The novel contributions of this paper are as follows:

- We introduce two algorithms that perform distributed inference for LDA models, one of which is simple to implement but does not necessarily sample from the correct posterior distribution, and the other which optimizes the correct posterior quantity but is more complex to implement and slower to run.
- We demonstrate that both distributed algorithms produce models that are statistically indistinguishable (in terms of predictive power) from models obtained on a single-processor, and they can learn these models much faster than using a single processor and only requiring storage of $\frac{1}{P}$th of the data on each processor.

## 2  Latent Dirichlet Allocation

Before introducing our distributed algorithms for LDA, we briefly review the standard LDA model. LDA models each of $D$ documents as a mixture over $K$ latent topics, each being a multinomial distribution over a $W$ word vocabulary. For document $j$, we first draw a mixing proportion $\theta_{k|j}$ from a Dirichlet with parameter $\alpha$. For the $i^{th}$ word in the document, a topic $z_{ij}$ is drawn with topic $k$ chosen with probability $\theta_{k|j}$, then word $x_{ij}$ is drawn from the $z_{ij}^{th}$ topic, with $x_{ij}$ taking on value $w$ with probability $\phi_{w|k}$. Finally, a Dirichlet prior with parameter $\beta$ is placed on the topics $\phi_{w|k}$. Thus, the generative process is given by

$$\theta_{k|j} \sim \mathcal{D}[\alpha] \quad \phi_{w|k} \sim \mathcal{D}[\beta] \quad z_{ij} \sim \theta_{k|j} \quad x_{ij} \sim \phi_{w|z_{ij}} \tag{1}$$

Given the observed words $\mathbf{x} = \{x_{ij}\}$, the task of Bayesian inference is to compute the posterior distribution over the latent topic indices $\mathbf{z} = \{z_{ij}\}$, the mixing proportions $\theta_{k|j}$, and the topics $\phi_{w|k}$. An efficient procedure is to use collapsed Gibbs sampling [5], where $\theta$ and $\phi$ are marginalized out, and the latent variables $\mathbf{z}$ are sampled. Given the current state of all but one variable $z_{ij}$, the conditional probability of $z_{ij}$ is

$$p(z_{ij} = k|\mathbf{z}^{\neg ij}, \mathbf{x}, \alpha, \beta) \propto (\alpha + n_{k|j}^{\neg ij})(\beta + n_{x_{ij}|k}^{\neg ij})(W\beta + n_k^{\neg ij})^{-1} \tag{2}$$

where the superscript $\neg ij$ means the corresponding data-item is excluded in the count values, and where $n_{jkw} = \#\{i : x_{ij} = w, z_{ij} = k\}$. We use the convention that missing indices are summed out: $n_{k|j} = \sum_w n_{jkw}$ and $n_{w|k} = \sum_j n_{jkw}$.

## 3  Distributed Inference Algorithms for LDA

We now present two versions of LDA where the data and the parameters are distributed over distinct processors. We distribute the $D$ documents over $P$ processors, with $D_P = \frac{D}{P}$ documents on each processor. We partition the data $\mathbf{x}$ (words from the $D$ documents) into $\mathbf{x} = \{\mathbf{x}_1, \ldots, \mathbf{x}_p, \ldots, \mathbf{x}_P\}$ and the corresponding topic assignments into $\mathbf{z} = \{\mathbf{z}_1, \ldots, \mathbf{z}_p, \ldots, \mathbf{z}_P\}$, where $\mathbf{x}_p$ and $\mathbf{z}_p$ only exist on processor $p$. Document-specific counts $n_{k|j}$ are likewise distributed, however every processor maintains its own copy of word-topic and topic counts, $n_{w|k}$ and $n_k$. We denote processor-specific counts as $n_{k|jp}, n_{w|kp}$ and $n_{kp}$.

### 3.1  Approximate Distributed Inference

In our Approximate Distributed LDA model (AD-LDA), we simply implement LDA on each processor, and simultaneous Gibbs sampling is performed independently on each of the $P$ processors, as if each processor thinks it is the only processor. On processor $p$, given the current state of all but one variable $z_{ijp}$, the topic assignment to the $i^{th}$ word in document $j$, $z_{ijp} \in \mathbf{z}_p$ is sampled from:

$$p(z_{ijp} = k|\mathbf{z}_p^{\neg ijp}, \mathbf{x}, \alpha, \beta) \propto (\alpha + n_{k|jp}^{\neg ijp})(\beta + n_{x_{ij}|kp}^{\neg ijp})(W\beta + n_{kp}^{\neg ijp})^{-1} \tag{3}$$

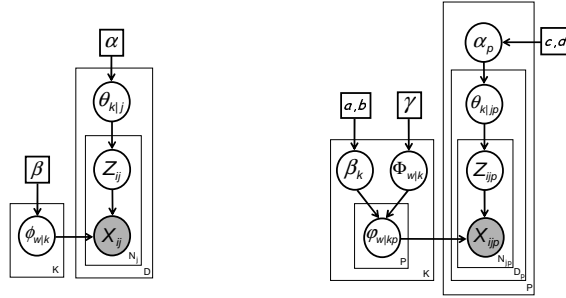

Figure 1: (Left) Graphical model for LDA. (Right) Graphical model for HD-LDA. Variables are repeated over the indices of the random variables. Square boxes indicate parameters.

Note that $n_{w|kp}$ is not the result of $P$ separate LDA models running on separate data. In particular $\sum_{w,k} n_{w|kp} = N$, where $N$ is the total number of words across all processors, as opposed to the number of words on processor $p$. After processor $p$ has reassigned $\mathbf{z}_p$, we have modified counts $n_{k|jp}$, $n_{w|kp}$, and $n_{kp}$. To merge back to a single set of counts, after a number of Gibbs sampling steps (e.g., after a single pass through the data on each processor) we perform the global update, using a reduce-scatter operation,

$$n_{w|k} \leftarrow n_{w|k} + \sum_p (n_{w|kp} - n_{w|k}), \qquad n_{w|kp} \leftarrow n_{w|k} \qquad (4)$$

where $n_{w|k}$ are the counts that all processors started with before the sweep of the Gibbs sampler. The counts $n_k$ are computed by $n_k = \sum_w n_{w|k}$. Note that this global update correctly reflects the topic assignments $\mathbf{z}$ (i.e., $n_{w|k}$ can also be regenerated using $\mathbf{z}$).

We can consider this algorithm to be an approximation to the single-processor Gibbs sampler in the following sense: at the start of each iteration, all of the processors have the same set of counts. However, as each processor starts sampling, the global count matrix is changing in a way that is unknown to each processor. Thus, in Equation 3, the sampling is not being done according to the true current global count (or true posterior distribution), but to an approximation. We have experimented with "repairing" reversibility of the sampler by adding a phase which re-traces the Gibbs moves starting at the (global) end-state, but we found that, due to the curse-of-dimensionality, virtually all steps ended up being rejected.

## 3.2   Hierarchical Distributed Inference

A more principled way to model parallel processes is to build them directly into the probabilistic model. Imagine a parent collection of topics $\Phi_{w|k}$. This parent has $P$ children $\varphi_{w|kp}$ which represent the topic distributions on the various processors. We assume $\varphi$ is sampled from $\Phi$ according to a Dirichlet distribution with topic-dependent strength parameter $\beta_k$. The model that lives on each processor is simply an LDA model. Hence, the generative process is given by,

$$\begin{aligned}
\Phi_{w|k} &\sim \mathcal{D}[\gamma] & \beta_k &\sim \mathcal{G}[a,b] & \alpha_p &\sim \mathcal{G}[c,d] \\
\varphi_{w|kp} &\sim \mathcal{D}[\beta_k \Phi_{w|k}] & \theta_{k|jp} &\sim \mathcal{D}[\alpha_p] \\
z_{ijp} &\sim \theta_{k|jp} & x_{ijp} &\sim \varphi_{w|z_{ijp}} & & & (5)
\end{aligned}$$

The graphical model corresponding to this Hierarchical Distributed LDA (HD-LDA) is shown on the right of Figure 1, with standard LDA shown on the left for comparison. This model is different than the two other topic hierarchies we found in the literature, namely 1) the deeper version of the hierarchical Dirichlet process mentioned in [6] and 2) Pachinko allocation [7]. The first places a deeper hierarchical prior on $\theta$ (instead of on $\varphi$) while the second deals with a document-specific hierarchy of topic-assignments. These types of hierarchies do not suit our need to facilitate parallel computation.

As is the case for LDA, inference for HD-LDA is most efficient if we marginalize out $\varphi$ and $\theta$. We derive the following conditional probabilities necessary for the Gibbs sampler,

$$p(z_{ijp} = k | \mathbf{z}_p^{\neg ijp}, \mathbf{x}, \alpha, \beta, \Phi) \propto (\alpha_p + n_{k|jp}^{\neg ijp})(\beta_k \Phi_{x_{ij}|k} + n_{x_{ij}|kp}^{\neg ijp})(\beta_k + n_{kp}^{\neg ijp})^{-1} \quad (6)$$

In our experiments we learn MAP estimates for the global variables $\Phi$, $\beta$ and $\alpha$. Alternatively, one can derive Gibbs sampling equations using the auxiliary variable method explained in [6], but we leave exploration of this inference technique for future research. Inference is thus based on integrating out $\theta$ and $\varphi$, sampling $\mathbf{z}$ and learning the MAP value of $\Phi$, $\beta$ and $\alpha$. The entire algorithm can be understood as expectation maximization on a collapsed space where the M-step corresponds to MAP-updates and the E-step corresponds to sampling. As such, the proposed Monte Carlo EM (MCEM) algorithm is guaranteed to converge in expectation (e.g., [8]). The MAP learning rules are derived by using the bounds derived in [9]. They are given by

$$\alpha_p \leftarrow \frac{c - 1 + \alpha_p \sum_{jk} \left[ \Psi(\alpha_p + n_{k|jp}) - \Psi(\alpha_p) \right]}{d + K \sum_j \left[ \Psi(K\alpha_p + n_{jp}) - \Psi(K\alpha_p) \right]}$$

$$\beta_k \leftarrow \frac{a - 1 + \beta_k \sum_{wp} \Phi_{w|k} \left[ \Psi(\beta_k \Phi_{w|k} + n_{w|kp}) - \Psi(\beta_k \Phi_{w|k}) \right]}{b + \sum_p \left[ \Psi(\beta_k + n_{kp}) - \Psi(\beta_k) \right]}$$

$$\Phi_{w|k} \leftarrow \frac{\gamma - 1 + \sum_p \beta_k \Phi_{w|k} \left[ \Psi(\beta_k \Phi_{w|k} + n_{w|kp}) - \Psi(\beta_k \Phi_{w|k}) \right]}{\gamma W - W + \sum_{wp} \beta_k \Phi_{w|k} \left[ \Psi(\beta_k \Phi_{w|k} + n_{w|kp}) - \Psi(\beta_k \Phi_{w|k}) \right]} \quad (7)$$

where $\Psi$ is the digamma function. Careful selection of hyper-parameters is critical to making HD-LDA work well, and we used our experience with AD-LDA to guide these choices. For AD-LDA $\sum_{w,k} n_{w|kp} = N$, but for HD-LDA $\sum_{w,k} n_{w|kp} \approx \frac{N}{P}$, so we choose $a$ and $b$ to make the mode of $\beta_k = \frac{P-1}{P} N$. We set $\gamma = 1 + \frac{1}{T}$. Finally we choose $c$ and $d$ to make the mode of $\alpha_p = 0.1$, matching the value of $\alpha$ used in our LDA and AD-LDA experiments.

We can view HD-LDA as a mixture model with $P$ LDA mixture components, where the data have been hard-assigned to their respective clusters (processors). The parameters of the clusters are generated from a shared prior distribution. This view clarifies the procedure we have adopted for testing: First we sample assignment variables $z_{ijp}$ for the first half of the test document (analogous to folding-in). Given these samples we compute the likelihood of the test document under the model for each processor. Assuming equal prior weights for each processor we then compute responsibilities, which are given by the likelihoods, normalized over processors. The probability of the remainder of the test document is then given by the responsibility-weighted average over the processors.

## 4   Experiments

The two distributed algorithms are initialized by first randomly assigning topics to $\mathbf{z}$, then from this counting topics in documents, $n_{k|jp}$, and words in topics, $n_{w|kp}$, for each processor. Recall for AD-LDA that the count arrays $n_{w|kp} = n_{w|k}$ are the same on every processor (initially, and after every global update). For each run of each algorithm, a sample was taken after 500 iterations of the Gibbs sampler, well after the typical burn-in period of 200-300 iterations. Multiple processors were simulated in software (by separating data, running sequentially through each processor, and simulating the global update step), except for the speedup experiments which were run on a 16-processor computer.

It is not obvious a priori that the AD-LDA algorithm will in general converge to a useful result. Later in this section we describe a set of systematic empirical results with AD-LDA, but we first use an illustrative toy example to provide some insight as to how AD-LDA learns a model. The toy example has $W = 3$ words, $K = 2$ topics. The left panel of Figure 2 shows the $L_1$ distance between the model's estimate of a particular topic-word distribution and the true distribution, as a function of Gibbs iterations, for both single-processor LDA and AD-LDA with $P = 2$. LDA and AD-LDA have qualitatively the same 3-phase learning dynamics[1]. The first 4 or so iterations ("early burn-in") correspond to somewhat random movement close to the randomly initialized starting point. In

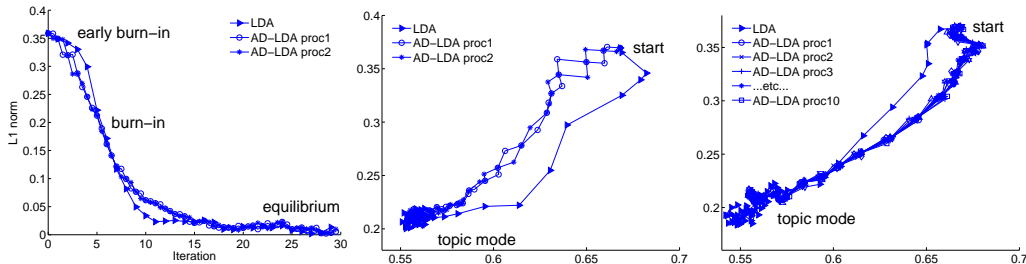

Figure 2: (Left) $L_1$ distance to the mode for LDA and for P=2 AD-LDA. (Center) Projection of topics onto simplex, showing convergence to mode. (Right) Same setup as center panel, but with $P = 10$ processors.

the next phase ("burn-in") both algorithms rapidly move in parameter space towards the posterior mode. And finally at equilibrium, both are sampling around the mode. The center panel of Figure 2 plots the same run, in the 2-d planar simplex corresponding to the 3-word topic distribution. This panel shows the paths in parameter space of each model, taking a few small steps near the starting point (top right corner), moving down to the true solution (bottom left), and then sampling near the posterior mode for the rest of the iterations. For each Gibbs iteration, the parameters corresponding to each of the two individual processors, and those parameters after merging, are shown (for AD-LDA). We observed that after the initial few iterations, the individual processor steps and the merge step each resulted in a move closer to the mode. The right panel in Figure 2 illustrates the same qualitative behavior as in the center panel, but now for 10 processors. One might worry that the AD-LDA algorithm would get "trapped" close to the initial starting point, e.g., due to repeated label mismatching of the topics across processors. In practice we have consistently observed that the algorithm quickly discards such configurations (due to the stochastic nature of the moves) and "latches" onto a consistent labeling that then rapidly moves it towards the posterior mode.

It is useful to think of LDA as an approximation to stochastic descent in the space of assignment variables $\mathbf{z}$. On a single processor, one can view Gibbs sampling during burn-in as a stochastic algorithm to move up the likelihood surface. With multiple processors, each processor computes an upward direction in its own subspace, keeping all other directions fixed. The global update step then recombines these directions by vector-addition, in the same way as one would compute a gradient using finite differences. This is expected to be accurate as long as the surface is locally convex or concave, but will break down at saddle-points. We conjecture AD-LDA works reliably because saddle points are 1) unstable and 2) rare due to the fact that the posterior appears often to be highly peaked for LDA models and high-dimensional count data sets.

To evaluate AD-LDA and HD-LDA systematically, we measured performance using test set perplexity, computed as $\text{Perp}(\mathbf{x}^{\text{test}}) = \exp(-\frac{1}{N^{\text{test}}} \log p(\mathbf{x}^{\text{test}}))$. For every test document, half the words (at random) are put in a fold-in part, and the remaining words are put in a test part. The document mix $\theta_{k|j}$ is learned using the fold-in part, and log probability is computed using this mix and words from the test part, ensuring that the test words are never seen before being used. For AD-LDA, the perplexity computation exactly follows that of LDA, since a single set of topic counts $n_{w|k}$ are saved when a sample is taken. In contrast, all $P$ copies of $n_{w|kp}$ are required to compute perplexity for HD-LDA, as described in the previous section. Except where stated, perplexities are computed for all algorithms using $S = 10$ samples from the posterior (from 10 different chains) using

$$\log p(\mathbf{x}^{\text{test}}) = \sum_{j,w} \log \frac{1}{S} \sum_s \sum_k \theta_{k|j}^s \phi_{w|k}^s \qquad \theta_{k|j}^s = \frac{\alpha + n_{k|j}^s}{K\alpha + n_j^s} \qquad \phi_{w|k}^s = \frac{\beta + n_{w|k}^s}{W\beta + n_k^s} \qquad (8)$$

with the analogous expression being used for HD-LDA.

We compared LDA (Gibbs sampling on a single processor) and our two distributed algorithms, AD-LDA and HD-LDA, using three data sets: KOS (from dailykos.com), NIPS (from books.nips.cc) and NYTIMES (from ldc.upenn.edu). Each data set was split into a training set and a test set. Size parameters for these data sets are shown in Table 1. For each corpus $W$ is the vocabulary size and $N$ is the total number of words. Using the three data sets and the three models we computed test set

|           | KOS     | NIPS      | NYTIMES     |
|-----------|---------|-----------|-------------|
| $D_{\text{train}}$ | 3000    | 1500      | 300,000     |
| $W$       | 6906    | 12,419    | 102,660     |
| $N$       | 410,000 | 1,900,000 | 100,000,000 |
| $D_{\text{test}}$  | 430     | 184       | 34,658      |

Table 1: Size parameters for the three data sets used in perplexity and speedup experiments.

perplexities for a range of topics $K$, and for number of processors, $P$, ranging from 10 to 1000 for our distributed models.

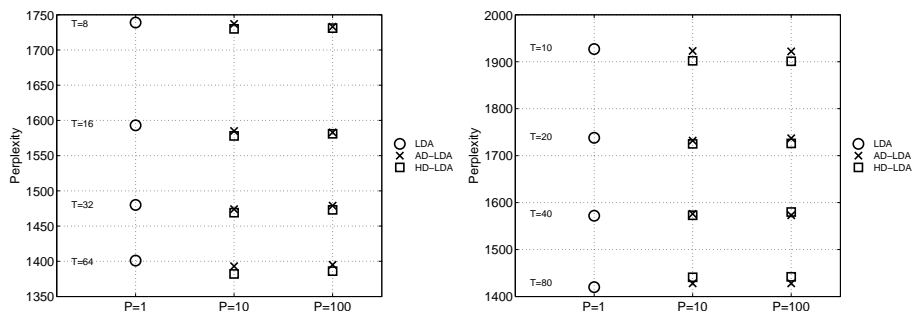

Figure 3: Test perplexity of models versus number of processors P for KOS (left) and NIPS (right). P=1 corresponds to LDA (circles), and AD-LDA (crosses), and HD-LDA (squares) are shown at P=10 and 100 .

Figure 3 clearly shows that, for a fixed number of topics, the perplexity results are essentially the same whether we use single-processor LDA or either of the two algorithms with data distributed across multiple processors (either 10 or 100). The figure shows the test set perplexity for KOS (left) and NIPS (right), versus number of processors, $P$. The $P = 1$ perplexity is computed by LDA (circles), and we use our distributed models – AD-LDA (crosses), and HD-LDA (squares) – to compute the $P = 10$ and $P = 100$ perplexities. Though not shown, perplexities for AD-LDA remained approximately constant as the number of processors was further increased to $P = 1000$ for KOS and $P = 500$ for NIPS, demonstrating effective distributed learning with only 3 documents on each processor. It is worth emphasizing that, despite no formal convergence guarantees, the approximate distributed algorithm converged to good solutions in every single one of the more than one thousand experiments we did using five real-world data sets, plus synthesized data sets designed to be "hard" to learn (i.e., topics mutually exclusively distributed over processors)—page limitations preclude a full description of all these results in this paper.

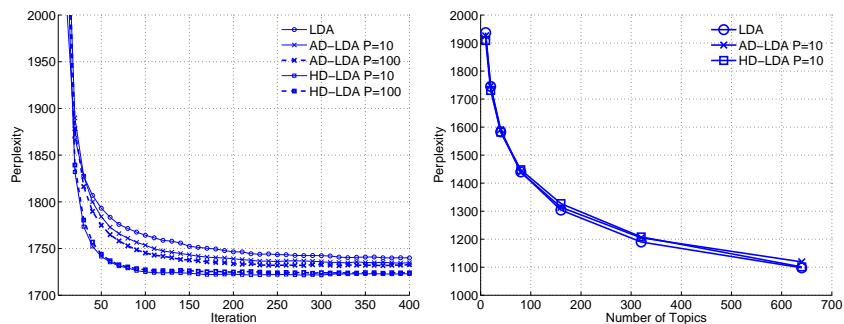

Figure 4: (Left) Test perplexity versus iteration.    (Right) Test perplexity versus number of topics.

To properly determine the utility of the distributed algorithms, it is necessary to check whether the parallelized samplers are systematically converging more slowly than single processor sampling. If

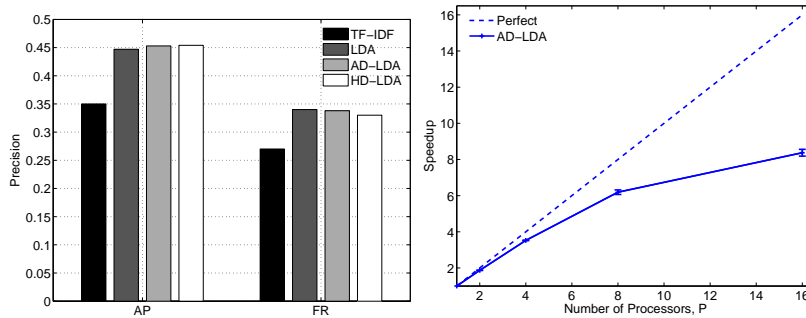

Figure 5: (Left) Precision/recall results.     (Right). Parallel speedup results.

this were the case, it would mitigate the computational gains of parallelization. In fact our experiments consistently showed (somewhat surprisingly) that the convergence rate for the distributed algorithms is just as rapid as for the single processor case. As an example, Figure 4 (left) shows test perplexity versus iteration number of the Gibbs sampler (NIPS, $K = 20$). During burn-in, up to iteration 200, the distributed models are actually converging slightly faster than single processor LDA. Also note that 1 iteration of AD-LDA (or HD-LDA) on a parallel computer takes a fraction of the wall-clock time of 1 iteration of LDA.

We also investigated whether the results were sensitive to the number of topics used in the models, e.g., perhaps the distributed algorithms' performance diverges when the number of topics becomes very large. Figure 4 (right) shows the test set perplexity computed on the NIPS data set using $S = 5$ samples, as a function of the number of topics, for the different algorithms and a fixed number of processors $P = 10$ (not shown here are the results for the KOS data set which were quite similar). The perplexities of the different algorithms closely track each other as $K$ varies. Sometimes the distributed algorithms produce slightly lower perplexities than those of single processor LDA. This lower perplexity may be due to: for AD-LDA, parameters constantly splitting and merging producing an internal averaging effect; and for HD-LDA, test perplexity being computed using $P$ copies of saved parameters.

Finally, to demonstrate that the low perplexities obtained from the distributed algorithms with $P = 100$ processors are not just due to averaging effects, we split the NIPS corpus into one hundred 15-document collections, and ran LDA separately on each of these hundred collections. Test perplexity ($K = 40$) computed by averaging 100-separate LDA models was 2117, versus the P=100 test perplexity of 1575 for AD-LDA and HD-LDA. This shows that simple averaging of results from separate processors does not perform nearly as well as the distributed coordinated learning.

Our distributed algorithms also perform well under other performance metrics. We performed precision/recall calculations using TREC's AP and FR collections and measured performance using the well-known mean average precision (MAP) metric used in IR research. Figure 5 (left) again shows that AD-LDA and HD-LDA (both using P=10) perform similarly to LDA. All three LDA models have significantly higher precision than TF-IDF on the AP and FR collections (significance was computed using a t-test at the 0.05 level). These calculations were run with $K = 200$.

The per-processor per-iteration time and space complexity of LDA and AD-LDA are shown in Table 2. AD-LDA's memory requirement scales well as collections grow, because while $N$ and $D$ can get arbitrarily large (which can be offset by increasing $P$), the vocabulary size $W$ asymptotes. Similarly the time complexity scales well since the leading order term $NK$ is divided by $P$. The $C$ term accounts for the communication cost of the reduce-scatter operation on the count difference $(n_{w|kp} - n_{w|k})$, which is executed in $\log P$ stages. Because of the additional $KW$ term, parallel efficiency will depend on $\frac{N}{PW}$, with increasing efficiency as this ratio increases. Space and time complexity of HD-LDA are similar to that of AD-LDA, but HD-LDA has bigger constants.

Using our large NYTIMES data set, we performed speedup experiments on a 16-processor SMP shared memory computer using $P = 1, 2, 4, 8$ and 16 processors (since we did not have access to a distributed memory computer). The single processor LDA run with 1000 iterations for this data set involves $10^{15}$ flops, and takes more than 10 days on a 3GHz workstation, so it is an ideal

| | LDA | AD-LDA |
|---|---|---|
| Space | $N + K(D + W)$ | $\frac{1}{P}(N + KD) + KW$ |
| Time | $NK$ | $\frac{1}{P}NK + KW + C$ |

Table 2: Space and time complexity of LDA and AD-LDA.

computation to speed up. The speedup results, shown in Figure 5 (right), show reasonable parallel efficiency, with a $8.5\times$ speedup using $P = 16$ processors. This speedup reduces our NYTIMES 10-day run (880 sec/iteration on 1 processor) to the order of 1 day (105 sec/iteration on 16 processors). Note, however, that while the implementation on an SMP machine captures some distributed effects (e.g. time to synchronize), it does not accurately reflect the extra time for communication. However, we do expect that for problems with large $\frac{N}{PW}$, parallel efficiency will be high.

## 5 Discussion and Conclusions

Prior work on parallelizing probabilistic learning algorithms has focused largely on EM-optimization algorithms, e.g., parallel updates of expected sufficient statistics for mixture models [2, 1]. In the statistical literature, the idea of running multiple MCMC chains in parallel is one approach to parallelization (e.g., the method of parallel tempering), but requires that each processor store a copy of the full data set. Since MCMC is inherently sequential, parallel sampling using distributed subsets of the data will not in general yield a proper MCMC sampler except in special cases [10]. Mimno and McCallum [11] recently proposed the DCM-LDA model, where processor-specific sets of topics are learned independently on each processor for local subsets of data, without any communication between processors, followed by a global clustering of the topics from the different processors. While this method is highly scalable, it does not lead to single global set of topics that represent individual documents, nor is it defined by a generative process.

We proposed two different approaches to distributing MCMC sampling across different processors for an LDA model. With AD-LDA we sample from an approximation to the posterior density by allowing different processors to concurrently sample latent topic assignments on their local subsets of the data. Despite having no formal convergence guarantees, AD-LDA works very well empirically and is easy to implement. With HD-LDA we adapt the underlying LDA model to map to the distributed computational infrastructure. While this model is more complicated than AD-LDA, and slower to run (because of digamma evaluations), it inherits the usual convergence properties of MCEM. Careful selection of hyper-parameters was critical to making HD-LDA work well.

In conclusion, both of our proposed algorithms learn models with predictive performance that is no different than single-processor LDA. On each processor they burn-in and converge at the same rate as LDA, yielding significant speedups in practice. The space and time complexity of both models make them scalable to run on enormous problems, for example, collections with billions to trillions of words. There are several potentially interesting research directions that can be pursued using the algorithms proposed here as a starting point, e.g., using asynchronous local communication (as opposed to the environment of synchronous global communications covered in this paper) and more complex schemes that allow data to adaptively move from one processor to another. The distributed scheme of AD-LDA can also be used to parallelize other machine learning algorithms. Using the same principles, we have implemented distributed versions of NMF and PLSA, and initial results suggest that these distributed algorithms also work well in practice.

## 6 Acknowledgements

This material is based upon work supported by the National Science Foundation: DN and PS were supported by NSF grants SCI-0225642, CNS-0551510, and IIS-0083489, AA was supported by an NSF graduate fellowship, and MW was supported by grants IIS-0535278 and IIS-0447903.

## Footnotes

[1]For clarity, the results in this figure are plotted for a single run, single data set, etc.—we observed qualitatively similar results over a large variety of such simulations

# References

[1] C. Chu, S. Kim, Y. Lin, Y. Yu, G. Bradski, A. Ng, and K. Olukotun. Map-Reduce for machine learning on multicore. In *NIPS 19*, pages 281–288. MIT Press, Cambridge, MA, 2007.

[2] W. Kowalczyk and N. Vlassis. Newscast EM. In *NIPS 17*, pages 713–720. MIT Press, Cambridge, MA, 2005.

[3] A. Das, M. Datar, A. Garg, and S. Rajaram. Google news personalization: Scalable online collaborative filtering. In *16th International World Wide Web Conference*, 2007.

[4] D. Blei, A. Ng, and M. Jordan. Latent Dirichlet allocation. *JMLR*, 3:993–1022, 2003.

[5] T. Griffiths and M. Steyvers. Finding scientific topics. In *Proceedings of the National Academy of Sciences*, volume 101, pages 5228–5235, 2004.

[6] Y.W. Teh, M. Jordan, M. Beal, and A. Blei. Sharing clusters among related groups: Hierarchical Dirichlet processes. In *NIPS 17*, pages 1385–1392. MIT Press, Cambridge, MA, 2005.

[7] W. Li and A. McCallum. Pachinko allocation: DAG-structured mixture models of topic correlations. In *ICML*, pages 577–584, 2006.

[8] G. Wei and M. Tanner. A Monte Carlo implementation of the EM algorithm and the poor man's data augmentation algorithms. *Journal of the American Statistical Association*, 85(411):699–704, 1990.

[9] T. Minka. Estimating a Dirichlet distribution. http://research.microsoft.com/ minka/papers/dirichlet/, 2003.

[10] A. Brockwell. Parallel markov chain monte carlo simulation by pre-fetching. In *J.Comp.Graph.Stats*, volume 15, pages 246–261, 2006.

[11] A. McCallum D. Mimno. Organizing the oca: Learning faceted subjects from a library of digital books. In *Joint Conference in Digital Libraries*, pages 376–385, 2007.

